# Speech Recognition using Connectionist Approaches

**Khalid Choukri**
SPRINT Coordinator
CAP GEMINI INNOVATION
118 rue de Tocqueville, 75017 Paris. France
e-mail: choukri@capsogeti.fr

# Abstract

This paper is a summary of SPRINT project aims and results. The project focus on the use of neuro-computing techniques to tackle various problems that remain unsolved in speech recognition. First results concern the use of feed-forward nets for phonetic units classification, isolated word recognition, and speaker adaptation.

## 1 INTRODUCTION

Speech is a complex phenomenon but it is useful to divide it into levels of representation. Connectionism paradigms and particularities are exploited to tackle the major problems in relationship with intra and inter speaker variabilities in order to improve the recognizer performance. For that purpose the project has been split into individual tasks which are depicted below:

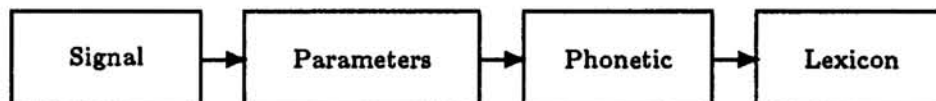

The work described herein concerns :

- Parameters-to-Phonetic: Classification of speech parameters using a set of "phonetic" symbols and extraction of speech features from signal.

- Parameters-to-Lexical: Classification of a sequence of feature vectors by lexical access (isolated word recognition) in various environments.

- Parameters-to-Parameters: Adaptation to new speakers and environments.

The following sections summarize the work carried out within this project. Details, including different nets description, are reported in the project deliverables (Choukri, 1990), (Bimbot, 1990), (Varga, 1990).

## 2  PARAMETERS-TO-PHONETIC

The objectives of this task were to assess various neural network topologies, and to examine the use of prior knowledge in improving results, in the process of acoustic-phonetic decoding of natural speech. These results were compared to classical pattern classification approaches such as k-nearest neighbour classifiers (K-nn), dynamic programming, and k-means.

### 2.1  DATABASES

The speech was uttered by one male speaker in French. Two databases were used: DB_1 made of isolated non-sense words (logatomes) which contains 6672 phonemes and DB_2 provided by the recording of 200 sentences which contains 5270 phonemes. DB_2 was split equally into training and test sets (2635 data each). 34 different labels were used : 1 per phoneme (not per allophone) and one for the silence. For each phoneme occurrence, 16 frames of signal (8 on each side of the label) were processed to provide a 16 Mel-scaled filter-bank vector.

### 2.2  CLASSICAL CLASSIFIERS

Experiments using k-NN and k-means classifiers were conducted to check the sufficient consistency of the data and to have some reference scores. A first protocol considered each pattern as a 256-dimension vector, and achieved k-nearest neighbours with the euclidean distance between references and tests. A second protocol attempted to decrease the time misalignments influences by carrying out some Dynamic Time Warping between references and tests and taking the sum of distances along the best path, as a distance measure between patterns. The same data was used in the framework of a k-means classifier, for various values of k (number of representatives per class). The best results are :

| Method | K-means (K $\geq$ 16) | K-nn (K=5) | K-nn + DTW (K=5) |
|--------|------------------------|------------|-------------------|
| Score  | 61.3 %                 | 72.2 %     | 77.5 %            |

### 2.3  NEURAL CLASSIFIERS
#### 2.3.1  LVQ Classifiers

Experiments were conducted using Learning Vector Quantization technique (LVQ) (Bennani, 1990). A study of the importance of the weights initialization procedure proved to be an important parameter for the classification performance. We have compared three initialization algorithms: k-means, LBG, Multiedit. With k-means and LBG, tests were conducted with different numbers of reference vectors, while for Multiedit, the algorithm discovers automatically representative vectors in the

training set, the number of which is therefore not specified in advance.

Initialization by LBG gave better performance for self-consistency (evaluation on the training database: DB_1) , whereas test performance on DB_2 (sentences) were similar for all procedures and very low. Further experiments were carried out on DB_2 both for training and testing. LBG initialization with 16 and 32 classes were tried (since they gave the best performances in the previous experiment). Even though the self-consistency for sentences is slightly lower than the one for logatomes, the improvement of recognition scores are far better as illustrated here:

| nb ref per class | 16 | 32 |
|---|---|---|
| K-means | 60.3 % | 61.3 % |
| LBG → LVQ | 62.4 % → 66.1 % | 63.2 % → 67.2 % |

This experiment and some others (not presented here) (Bimbot, 1990) confirm that the failure of previous experiments is more due to a mismatch between the corpora for this recognition method, than an inadequacy of the classification technique itself.

### 2.3.2   The Time-Delay Neural Network (TDNN) Classifiers

A TDNN, as introduced by A. Waibel (Waibel, 1987), can be described by its set of typological parameters, i.e. :

$$M_0 \times N_0 \ / \ P_0, S_0 \ - \ M_1 \times N_1 \ / \ P_1, S_1 \ - \ M_2 \times N_2 \ - \ K \times 1 \ .$$

In the following a "TDNN-derived" network has a similar architecture, except that $M_2$ is not constrained to be equal to K, and the connectivity between the last 2 layers is full. Various TDNN-derived architectures were tested on recognizing phonemes from sentences (DB_2) after learning on the logatomes (DB_1). Best results are given below:

| TDNN-derived structure | self-consist. | reco score |
|---|---|---|
| 16x16 / 2,1 - 8x15 / 7,2 - 5x5 - 34x1 | 63.9 % | 48.1 % |
| 16x16 / 2,1 - 16x15 / 7,4 - 11x3 - 34x1 | 75.1 % | 54.8 % |
| 16x16 / 4,1 - 16x13 / 5,2 - 16x5 - 34x1 | 81.0 % | 60.5 % |
| 16x16 / 2,1 - 16x15 / 7,4 - 16x3 - 34x1 | 79.8 % | 60.8 % |

The first net is clearly not powerful enough for the task, so the number of free parameters has be increased. This upgraded the results immediately as can be seen for the other nets. The third and fourth nets have equivalent performance, they differ in the local windows width and delays. Other tested architectures did not increase this performance. The main difference between training and test sets is certainly the different speaking rate, and therefore the existence of important time distorsions. Though TDNN-derived architectures seem more able to handle this kind of distorsions than LVQ, as the generalization performance is significantly higher for similar learning self-consistency, but both fail to remove all temporel misalignment

effects.

In order to upgrade classification performance we changed the cost function which is minimized by the network : the error term corresponding to the desired output is multiplied by a constant H superior to 1, the terms of the error corresponding to other outputs being left unchanged to compensate the deficiency of the simple mean square error procedure. We obtained our best results with the best TDDN-derived net we experimented for H=2 :

| Database | Net : | self-consist. | reco score |
|----------|-------|---------------|------------|
| DB_1 | 16x16 / 4,1 - 16x13 / 5,2 - 16x5 - 34x1 | 87.0 % | 63.0 % |
| DB_2 | 16x16 / 4,1 - 16x13 / 5,2 - 16x5 - 34x1 | 87.0 % | 78.0 % |

The too small number of independent weights (too low-dimensioned TDNN-derived architecture) makes the problem too constrained. A well chosen TDNN-derived architecture can perform as well as the best k-nearest neighbours strategy. Performance gets lower for data that mainly differ by a significant speaking rate mismatch which could indicate that TDNN-derived architectures do not manage to handle all kinds of time distortions. So it is encouraging to combine different networks and classical methods to deal with the temporal and sequential aspects of speech.

### 2.3.3   Combination of TDNN and LVQ

A set of experiments using a combined TDNN-derived network and LVQ architecture were conducted. For these experiments, we have used the best nets found in previous experiments. The main parameter of these experiments is the number of hidden cells in the last layer of the TDNN-derived network which is the input layer of LVQ (Bennani, 1990).

Evaluation on DB_1 with various numbers of references per class gave the following recognition scores:

| refs per class | 4 | 8 | 16 |
|----------------|-----|-----|-----|
| TDNN +k-means | 76.2 % | 78.1 % | 79.8 % |
| TDNN +LBG | 77.7 % | 79.9 % | 81.3 % |
| TDNN +LVQ (LBG for initialization) | 78.4 % | 82.1 % | 81.4 % |

Best results have been obtained with 8 references per class and the LBG algorithm to initialize the LVQ module. The best performance on the test set (82.1 % ) represents a significant increase (4 % ) compared to the best TDNN-derived network.

Other experiments were performed on TDNN + LVQ by using a modified LVQ architecture, presented in (Bennani, 1990), which is an extension of LVQ built to automatically weight the variables according to their importance for the classification. We obtain a recognition score of 83.6 % on DB_2 (training and tests on sentences).

We also used low dimensioned TDNNs for discriminating between phonetic features (Bimbot, 1990), assuming that phonetics will provide a description of speech that will appropriately constrain a priori a neural network, the TDNN structure war-

ranting the desirable property of shift invariance.

The feature extraction approach can be considered as an other way to use prior knowledge for solving a complex problem with neural networks. The results obtained in these experiments are an interesting starting point for designing a large modular network where each module is in charge of a simple task, directly related to a well-defined linguistic phenomenon (Bimbot, 1990).

## 2.4   CONCLUSIONS

Experiments with LVQ alone, a TDNN-derived network alone and combined TDNN-LVQ architectures proved the combined architecture to be the most efficient with respect to our databases as summarized below (training and tests on DB_2):

| k-means | LVQ | k-nn | k-nn + DTW | TDNN | TDNN + LVQ |
|---------|-----|------|------------|------|------------|
| 61.3 % | 67.2 % | 72.2 % | 77.5 % | 78.0 % | 83.6 % |

# 3   PARAMETERS-TO-LEXICAL

The main objective of this task is to use neural nets for the classification of a sequence of speech frames into lexical items (isolated words). Many factors affect the performance of automatic speech recognition systems. They have been categorized into those relating to speaker independent recognition mode, the time evolution of speech (time representation of the neural network input), and the effects of noise . The two first topics are described herein while the third one is described in (Varga, 1990).

## 3.1   USE OF VARIOUS NETWORK TOPOLOGIES

Experiments were carried out to examine the performance of several network topologies such as those evaluated in section 2. A TDNN can be thought of as a single Hidden Markov Model state spread out in time. The lower levels of the network are forced to be shift-invariant, and instantiate the idea that the absolute time of an event is not important. Scaly networks are similar to TDDNs in that the hidden units of a scaly network are fed by partially overlapping input windows. As reported in previous sections, LVQ proved to be efficient for the phoneme classification task and an "optimal" architecture was found as a combination of a TDNN and LVQ. It was used herein for isolated word recognition.

From experiments reported in detail in (Varga, 1990) there seems little justification for fully-connected networks with their thousands of weights when TDNNs and Scaly networks with hundreds of weights have very similar performance. This performance is about **83%** (the nearest class mean classifier gave a performance of 69%) on the E-set database (a portion of the larger CONNEX alphabet database which British Telecom Research Laboratories have prepared for experiments on neural networks). The first utterance by each speaker of the "E" words: "B, C, D,

E, G, P, T, V" were used. The database is divided into training and test sets, each consisting of approximately 400 words and 50 speakers.

Other experiments were conducted on an isolated digits recognition task, speaker independent mode (25 speakers for training and 15 for test), using networks already introduced. A summary of the best performance obtained is:

| K-means | | TDNN | | LVQ | | TDNN+LVQ | |
|---|---|---|---|---|---|---|---|
| train. | test | train. | test | train. | test | train. | test |
| 97.38 | 90.57 | 98,90 | 94.0 | 98.26 | 92.57 | 99.90 | 97.50 |

Performance for training is roughly equivalent for all algorithms. For generalization, performance of the combined architecture is clearly superior to other techniques.

## 3.2  TIME EVOLUTION OF SPEECH

In contrast to images as patterns of specific size, speech signals display a temporal evolution. Approaches have to be developed on how a network with its fixed number of input units can cover word patterns of variable size and also account for the dynamic time variations within words.

Different projections onto the fixed-size collection of N×M network input elements (number of vectors × number of coefficients per vector) have been tested, such as :

**Linear Normalization** : the boundaries of a word are determined by a conventional endpoint detection algorithm and the N' feature vectors linearly compressed or expanded to N by averaging or duplicating vectors,

**Time Warp** : word boundaries are located initially. Some parts of a word of length N' are compressed, while others are stretched and some remain constant with respect to speech characteristics,

**Noise Boundaries** : the sequence of N' vectors of a word are placed in the middle of or at random within the area of the desired N vectors and the margins padded with the noise in the speech pauses,

**Trace Segmentation** : the procedure essentially involves the division of the trace that is followed by the temporal course in the M-dimensional feature vector space, into a constant number of new sections of identical length.

These time normalization procedures were used with the scaly neural network (Varga, 1990). It turned out that three methods for time representation - time normalization, trace segmentation with endpoint detection or with noise boundaries - are well suited to solve the transformation problem for a fixed input network layer. The recognition scores are in the 98.5% range (with±1% deviation) for 10 digits and 99.5% for a 57 words in speaker independent mode. There is no clear indication that one of these approaches is superior to the other ones.

## 3.3  CONCLUSIONS

The neural network techniques investigated have delivered comparable performance to classical techniques. It is now well agreed that Hybrid systems (Integration of

Hidden Markov Modeling and MLPs) yield enhanced performance. Initial steps have been made towards the integration of Hidden Markov Models and MLPs. Mathematical formulations are required to unify hybrid models. The temporal aspect of speech has to be carefully considered and taken into account by the formalism.

# 4    PARAMETERS-TO-PARAMETERS

The main objective of this task was to provide the speech recognizer with a set of parameters adapted to the current user without any training phase.

Spectral parameters corresponding to the same sound uttered by two speakers are generally different. Speaker-independent recognizers usually take this variability into account, using stochastic models and/or multi-references. An alternative approach consists in learning spectral mappings to transform the original set of parameters into another one more adapted with respect to the characteristics of the current user and the speech acquisition conditions. The way to proceed can be summed up as follows :

- Load of the standard dictionary of the reference speaker,

- Acquisition of an adaptation vocabulary for the new speaker,

- Each new utterance is time-warped against the corresponding reference utterance. Thus temporal variability is softened and corresponding feature vectors are available (input-output pairs),

- The spectral transformations are learned from these associated vectors,

- The adaptation operator is applied to the reference dictionary, leading to an adapted one,

- The recognizer is evaluated using the obtained adapted dictionary.

The mathematical formulation is based on a very important result, regarding input-output mappings, and demonstrated by Funahashi (Funahashi, 1989) and Hornik, Stinchcombe & White (Hornik, 1989). They proved that a network using a single hidden layer (a net with 3 layers) with an arbitrary squashing function can approximate any Borel measurable function to any desired degree of accuracy.

Experiments were conducted (see details in (Choukri, 1990)) on a speech isolated word database consisting of 20 English words recorded 26 times by 16 different speakers (TI data base (Choukri, 1987)). The first repetition of the 20 words are reference templates, tests are conducted on the remaining 25 repetitions. Before adaptation, the cross-speaker scores is of 68%. On the average adaptation with the multi-layer perceptron provides a 15% improvement compared to the non-adapted results.

# 5   CONCLUSIONS

For phonetic classifications, sophisticated networks, combinations of TDNNs and LVQ, revealed to be more efficient than classical approaches or simple network architectures; their use for isolated word recognition offered comparable performance. Various approaches to cope with temporal distortions were implemented and demonstrate that combination of sophisticated neural networks and their cooperation with HMM is a promising research axis. It has also been established that basic MLPs are efficient tools to learn speaker-to-speaker mappings for speaker adaptation procedures. We are expecting more sophisticated MLPs (recurrent and context sensitive) to perform better.

Acknowledgements:

This project is partially supported by the European ESPRIT Basic research Actions programme (BRA 3228). The partners involved are: CGInn (F), ENST (F), IRIAC (F), RSRE (UK), SEL (FRG), and UPM (SPAIN).

References

K. Choukri. (1990) *Speech processing and recognition using integrated neurocomputing techniques: ESPRIT Project SPRINT (Bra 3228), First deliverable of Task 2*, June 1990.

F. Bimbot. (1990) *Speech processing and recognition using integrated neurocomputing techniques: ESPRIT project SPRINT (Bra 3228), First deliverable of task 3*, June 1990.

A. Varga. (1990) *Speech processing and recognition using integrated neurocomputing techniques: ESPRIT Project SPRINT (Bra 3228), First deliverable of Task 5*, June 1990.

A. Waibel, T. Hanazawa, G. Hinton, K. Shikano, and K. Lang. (1987) *Phoneme recognition using Time-Delay Neural Networks.*, Technical Report, CMU / ATR, Oct 30, 1987.

Y. Bennani, N. Chaourar, P. Gallinari, and A. Mellouk. (1990) *Comparison of Neural Net models on speech recognition tasks*, Technical Report, Universit of Paris Sud, LRI, 1990.

Ken-Ichi Funahashi. (1989) *On the approximate realization of continuous mappings by neural networks*, in Neural Networks, 2(2):183–192, march 1989.

K. Hornik, M. Stinchcombe, and H. White. (1989) *Multilayer feedforward networks are universal approximators.*, in Neural Networks, vol. 2(number 5):359–366, 1989.

K. Choukri. (1987) *Several approaches to Speaker Adaptation in Automatic Speech Recognition Systems*, PhD thesis, ENST (Télécom Paris), Paris, 1987.

---

**AUTHORS AND CONTRIBUTORS**

| | | | |
|---|---|---|---|
| Y. BENNANI | F. BIMBOT | J. BRIDLE | N. CHAOURAR |
| K. CHOUKRI | L. DODD | F. FOGELMAN | P. GALLINARI |
| D. HOWELL | M. IMMENDORFER | A. KRAUSE | K. McNAUGHT |
| A. MELLOUK | C. MONTACIE | R. MOORE | O. SEGARD |
| H. VALBRET | A. VARGA | A. WALLYN | |

---
